# Natural Policy Gradient Methods with Parameter-based Exploration for Control Tasks

**Atsushi Miyamae**[†‡]**, Yuichi Nagata**[†]**, Isao Ono**[†]**, Shigenobu Kobayashi**[†]
†: Department of Computational Intelligence and Systems Science
Tokyo Institute of Technology, Kanagawa, Japan
‡: Research Fellow of the Japan Society for the Promotion of Science
{miyamae@fe., nagata@fe., isao@, kobayasi@}dis.titech.ac.jp

## Abstract

In this paper, we propose an efficient algorithm for estimating the natural policy gradient using parameter-based exploration; this algorithm samples directly in the parameter space. Unlike previous methods based on natural gradients, our algorithm calculates the natural policy gradient using the inverse of the exact Fisher information matrix. The computational cost of this algorithm is equal to that of conventional policy gradients whereas previous natural policy gradient methods have a prohibitive computational cost. Experimental results show that the proposed method outperforms several policy gradient methods.

## 1   Introduction

Reinforcement learning can be used to handle policy search problems in unknown environments. Policy gradient methods [22, 20, 5] train parameterized stochastic policies by climbing the gradient of the average reward. The advantage of such methods is that one can easily deal with continuous state-action and continuing (not episodic) tasks. Policy gradient methods have thus been successfully applied to several practical tasks [11, 21, 16].

In the domain of control, a policy is often constructed with a controller and an exploration strategy. The controller is represented by a domain-appropriate pre-structured parametric function. The exploration strategy is required to seek the parameters of the controller. Instead of directly perturbing the parameters of the controller, conventional exploration strategies perturb the resulting control signal. However, a significant problem with the sampling strategy is that the high variance in their gradient estimates leads to slow convergence. Recently, *parameter-based exploration* [18] strategies that search the controller parameter space by direct parameter perturbation have been proposed, and these have been demonstrated to work more efficiently than conventional strategies [17, 18, 13]. Another approach to speeding up policy gradient methods is to replace the gradient with the *natural* gradient [2], the so-called natural policy gradient [9, 4, 15]; this is motivated by the intuition that a change in the policy parameterization should not influence the result of the policy update. The combination of parameter-based exploration strategies and the natural policy gradient is expected to result in improvements in the convergence rate; however, such an algorithm has not yet been proposed.

However, natural policy gradients with parameter-based exploration strategies have a disadvantage in that the computational cost is high. The natural policy gradient requires the computation of the inverse of the Fisher information matrix (FIM) of the policy distribution; this is prohibitively expensive, especially for a high-dimensional policy. Unfortunately, parameter-based exploration strategies tend to have higher dimensions than control-based ones. Therefore, the expected method is difficult to apply for realistic control tasks.

In this paper, we propose a new reinforcement learning method that combines the natural policy gradient and parameter-based exploration. We derive an efficient algorithm for estimating the natural policy gradient with a particular exploration strategy implementation. Our algorithm calculates the natural policy gradient using the inverse of the exact FIM and the Monte Carlo-estimated gradient. The resulting algorithm, called *natural policy gradients with parameter-based exploration* (NPGPE), has a computational cost similar to that of conventional policy gradient algorithms. Numerical experiments show that the proposed method outperforms several policy gradient methods, including the current state-of-the-art NAC [15] with control-based exploration.

## 2  Policy Search Framework

We consider the standard reinforcement learning framework in which an agent interacts with a Markov decision process. In this section, we review the estimation of policy gradients and describe the difference between control- and parameter-based exploration.

### 2.1  Markov Decision Process Notation

At each discrete time $t$, the agent observes state $\mathbf{s}_t \in \mathcal{S}$, selects action $\mathbf{a}_t \in \mathcal{A}$, and then receives an instantaneous reward $r_t \in \Re$ resulting from a state transition in the environment. The state $\mathcal{S}$ and the action $\mathcal{A}$ are both defined as continuous spaces in this paper. The next state $\mathbf{s}_{t+1}$ is chosen according to the transition probability $p_T(\mathbf{s}_{t+1}|\mathbf{s}_t, \mathbf{a}_t)$, and the reward $r_t$ is given randomly according to the expectation $\mathcal{R}(\mathbf{s}_t, \mathbf{a}_t)$. The agent does not know $p_T(\mathbf{s}_{t+1}|\mathbf{s}_t, \mathbf{a}_t)$ and $\mathcal{R}(\mathbf{s}_t, \mathbf{a}_t)$ in advance.

The objective of the reinforcement learning agent is to construct a policy that maximizes the agent's performance. A parameterized policy $\pi(\mathbf{a}|\mathbf{s}, \theta)$ is defined as a probability distribution over an action space under a given state with parameters $\theta$. We assume that each $\theta \in \Re^d$ has a unique well-defined stationary distribution $p_D(\mathbf{s}|\theta)$. Under this assumption, a natural performance measure for infinite horizon tasks is the *average reward*

$$\eta(\theta) = \int_{\mathcal{S}} p_D(\mathbf{s}|\theta) \int_{\mathcal{A}} \pi(\mathbf{a}|\mathbf{s}, \theta)\mathcal{R}(\mathbf{s}, \mathbf{a})d\mathbf{a}d\mathbf{s}.$$

### 2.2  Policy Gradients

Policy gradient methods update policies by estimating the gradient of the average reward w.r.t. the policy parameters. The state-action value is $Q_\theta(\mathbf{s}, \mathbf{a}) = E[\sum_{t=1}^{\infty} r_t - \eta(\theta)|\mathbf{s}_1 = \mathbf{s}, \mathbf{a}_1 = \mathbf{a}, \theta]$, and it is assumed that $\pi(\mathbf{a}|\mathbf{s}, \theta)$ is differentiable w.r.t. $\theta$. The exact gradient of the average reward (see [20]) is given by

$$\nabla_\theta \eta(\theta) = \int_{\mathcal{S}} p_D(\mathbf{s}|\theta) \int_{\mathcal{A}} \pi(\mathbf{a}|\mathbf{s}, \theta)\nabla_\theta \log \pi(\mathbf{a}|\mathbf{s}, \theta)Q_\theta(\mathbf{s}, \mathbf{a})d\mathbf{a}d\mathbf{s}. \tag{1}$$

The natural gradient [2] has a basis in information geometry, which studies the Riemannian geometric structure of the manifold of probability distributions. A result in information geometry states that the FIM defines a Riemannian metric tensor on the space of probability distributions [3] and that the direction of the steepest descent on a Riemannian manifold is given by the natural gradient, given by the conventional gradient premultiplied by the inverse matrix of the Riemannian metric tensor [2]. Thus, the natural gradient can be computed from the gradient and the FIM, and it tends to converge faster than the conventional gradient.

Kakade [9] applied the natural gradient to policy search; this was called as the natural policy gradient. If the FIM is invertible, the natural policy gradient $\tilde{\nabla}_\theta \eta(\theta) \equiv \mathbf{F}_\theta^{-1} \nabla_\theta \eta(\theta)$ is given by the policy gradient premultiplied by the inverse matrix of the FIM $\mathbf{F}_\theta$. In this paper, we employ the FIM proposed by Kakade [9], defined as

$$\mathbf{F}_\theta = \int_{\mathcal{S}} p_D(\mathbf{s}|\theta) \int_{\mathcal{A}} \pi(\mathbf{a}|\mathbf{s}, \theta)\nabla_\theta \log \pi(\mathbf{a}|\mathbf{s}, \theta)\nabla_\theta \log \pi(\mathbf{a}|\mathbf{s}, \theta)^{\mathrm{T}}d\mathbf{a}d\mathbf{s}.$$

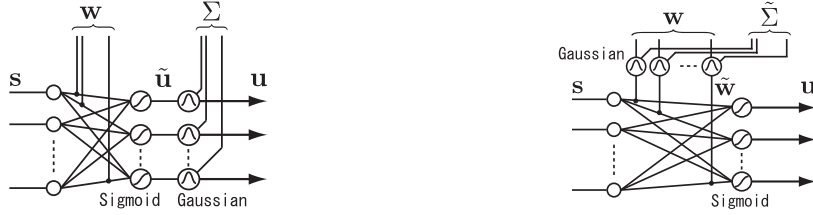

Figure 1: Illustration of the main difference between control-based exploration and parameter-based exploration. The controller $\psi(\mathbf{u}|\mathbf{s}, \mathbf{w})$ is represented by a single-layer perceptron. While the control-based exploration strategy (left) perturbs the resulting control signal, the parameter-based exploration strategy (right) perturbs the parameters of the controller.

## 2.3 Learning from Samples

The calculation of (1) requires knowledge of the underlying transition probabilities $p_D(\mathbf{s}|\theta)$. The GPOMDP algorithm [5] instead computes a Monte Carlo approximation of (1): the agent interacts with the environment, producing an observation, action, and reward sequence $\{\mathbf{s}_1, \mathbf{a}_1, r_1, \mathbf{s}_2, ..., \mathbf{s}_T, \mathbf{a}_T, r_T\}$. Under mild technical assumptions, the policy gradient approximation is

$$\nabla_\theta \eta(\theta) \approx \frac{1}{T} \sum_{t=1}^{T} r_t \mathbf{z}_t,$$

where $\mathbf{z}_t = \beta \mathbf{z}_{t-1} + \nabla_\theta \log \pi(\mathbf{a}_t|\mathbf{s}_t, \theta)$ is called the *eligibility trace* [12], $\nabla_\theta \log \pi(\mathbf{a}_t|\mathbf{s}_t, \theta)$ is called the *characteristic eligibility* [22], and $\beta$ denotes the discount factor ($0 \le \beta < 1$). As $\beta \to 1$, the estimation approaches the true gradient [1] , but the variance increases ($\beta$ is set to 0.9 in all experiments). We define $\tilde{\nabla}_\theta \log \pi(\mathbf{a}_t|\mathbf{s}_t, \theta) \equiv \mathbf{F}_\theta^{-1} \nabla_\theta \log \pi(\mathbf{a}_t|\mathbf{s}_t, \theta)$. Therefore, the natural policy gradient approximation is

$$\tilde{\nabla}_\theta \eta(\theta) \approx \frac{1}{T} \sum_{t=1}^{T} \mathbf{F}_\theta^{-1} r_t \mathbf{z}_t = \frac{1}{T} \sum_{t=1}^{T} r_t \tilde{\mathbf{z}}_t, \tag{2}$$

where $\tilde{\mathbf{z}}_t = \beta \tilde{\mathbf{z}}_{t-1} + \tilde{\nabla}_\theta \log \pi(\mathbf{a}_t|\mathbf{s}_t, \theta)$. To estimate the natural policy gradient, the heuristic suggested by Kakade [9] used

$$\mathbf{F}_{\theta,t} = (1 - \frac{1}{t})\mathbf{F}_{\theta,t-1} + \frac{1}{t}(\nabla_\theta \log \pi(\mathbf{a}_t|\mathbf{s}_t, \theta) \nabla_\theta \log \pi(\mathbf{a}_t|\mathbf{s}_t, \theta)^{\mathrm{T}} + \lambda \mathbf{I}), \tag{3}$$

the online estimate of the FIM, where $\lambda$ is a small positive constant.

## 2.4 Parameter-based Exploration

In most control tasks, we attempt to have a (deterministic or stochastic) controller $\psi(\mathbf{u}|\mathbf{s}, \mathbf{w})$ and an exploration strategy, where $\mathbf{u} \in \mathcal{U} \subseteq \Re^m$ denotes control and $\mathbf{w} \in \mathcal{W} \subseteq \Re^n$, the parameters of the controller. The objective of learning is to seek suitable values of the parameters $\mathbf{w}$, and the exploration strategy is required to carry out stochastic sampling near the current parameters. A typical exploration strategy model, we call *control-based exploration*, would be a normal distribution for the control space (Figure1 (left)). In this case, the action of the agent is control, and the policy is represented by

$$\pi_U(\mathbf{u}|\mathbf{s}, \theta) = \frac{1}{(2\pi)^{m/2}|\Sigma|^{1/2}} \exp\left(-\frac{1}{2}(\mathbf{u} - \psi(\mathbf{s}, \mathbf{w}))^{\mathrm{T}} \Sigma^{-1}(\mathbf{u} - \psi(\mathbf{s}, \mathbf{w}))\right) : \mathcal{S} \to \mathcal{U},$$

where $\Sigma$ is the $m \times m$ covariance matrix and the agent seeks $\theta = \langle \mathbf{w}, \Sigma \rangle$. The control at time $t$ is generated by

$$\tilde{\mathbf{u}}_t = \psi(\mathbf{s}_t, \mathbf{w}),$$
$$\mathbf{u}_t \sim \mathcal{N}(\tilde{\mathbf{u}}_t, \Sigma).$$

One useful feature of such a *Gaussian unit* [22] is that the agent can potentially control its degree of exploratory behavior.

The control-based exploration strategy samples near the output of the controller. However, the structures of the parameter space and the control space are not always identical. Therefore, the sampling strategy generates controls that are not likely to be generated from the current controller, even if the exploration variances decrease. This property leads to large variance gradient estimates. This might be one reason why the policy improvement gets stuck.

To address this issue, Sehnke et al. [18] introduced a different exploration strategy for policy gradient methods called *policy gradients with parameter-based exploration* (PGPE). In this approach, the action of the agent is the parameters of the controller, and the policy is represented by

$$\pi_W(\tilde{\mathbf{w}}|\mathbf{s}, \theta) = \frac{1}{(2\pi)^{n/2}|\tilde{\Sigma}|^{1/2}} \exp\left(-\frac{1}{2}(\tilde{\mathbf{w}} - \mathbf{w})^{\mathrm{T}}\tilde{\Sigma}^{-1}(\tilde{\mathbf{w}} - \mathbf{w})\right) : \mathcal{S} \to \mathcal{W},$$

where $\tilde{\Sigma}$ is the $n \times n$ covariance matrix and the agent seeks $\theta = \langle \mathbf{w}, \tilde{\Sigma} \rangle$. The controller is included in the dynamics of the environment, and the control at time $t$ is generated by

$$\tilde{\mathbf{w}}_t \sim \mathcal{N}(\mathbf{w}, \tilde{\Sigma}),$$
$$\mathbf{u}_t = \psi(\mathbf{s}_t, \tilde{\mathbf{w}}_t).$$

GPOMDP-based methods can estimate policy gradients such as partially observable settings, i.e., the policy $\pi_W(\tilde{\mathbf{w}}|\mathbf{s}, \theta)$ excludes the observation of the current state. Because this exploration strategy directly perturbs the parameters (Figure1 (right)), the samples are generated near the current parameters under small exploration variances. Note that the advantage of this framework is that because the gradient is estimated directly by sampling the parameters of the controller, the implementation of the policy gradient algorithms does not require $\frac{\partial}{\partial \theta}\psi$, which is difficult to derive from complex controllers.

Sehnke et al. [18] demonstrated that PGPE can yield faster convergence than the control-based exploration strategy in several challenging episodic tasks. However, the parameter-based exploration tends to have a higher dimension than the control-based one. Therefore, because of the computational cost of the inverse of $\mathbf{F}_\theta$ calculated by (3), natural policy gradients find limited applications.

# 3 Natural Policy Gradients with Parameter-based Exploration

In this section, we propose a new algorithm called *natural policy gradients with parameter-based exploration* (NPGPE) for the efficient estimation of the natural policy gradient.

## 3.1 Implementation of Gaussian-based Exploration Strategy

We employ the policy representation model $\mu(\tilde{\mathbf{w}}|\theta)$, a multivariate normal distribution with parameters $\theta = \langle \mathbf{w}, \mathbf{C} \rangle$, where $\mathbf{w}$ represents the mean and $\mathbf{C}$, the Cholesky decomposition of the covariance matrix $\tilde{\Sigma}$ such that $\mathbf{C}$ is an $n \times n$ upper triangular matrix and $\tilde{\Sigma} = \mathbf{C}^{\mathrm{T}}\mathbf{C}$. Sun et al. [19] noted two advantages of this implementation: $\mathbf{C}$ makes explicit the $n(n+1)/2$ independent parameters determining the covariance matrix $\tilde{\Sigma}$; in addition, the diagonal elements of $\mathbf{C}$ are the square roots of the eigenvalues of $\tilde{\Sigma}$, and therefore, $\mathbf{C}^{\mathrm{T}}\mathbf{C}$ is always positive semidefinite. In the remainder of the text, we consider $\theta$ to be an $[n(n+3)/2]$-dimensional column vector consisting of the elements of $\mathbf{w}$ and the upper-right elements of $\mathbf{C}$, i.e.,

$$\theta = [\mathbf{w}^{\mathrm{T}}, (\mathbf{C}_{1:n,1})^{\mathrm{T}}, (\mathbf{C}_{2:n,2})^{\mathrm{T}}, ..., (\mathbf{C}_{n:n,n})^{\mathrm{T}}]^{\mathrm{T}}.$$

Here, $\mathbf{C}_{k:n,k}$ is the sub-matrix in $\mathbf{C}$ at row $k$ to $n$ and column $k$.

## 3.2 Inverse of Fisher Information Matrix

Previous natural policy gradient methods [9] use the empirical FIM, which is estimated from a sample path. Such methods are highly inefficient for $\mu(\tilde{\mathbf{w}}|\theta)$ to invert the empirical FIM, a matrix with $O(n^4)$ elements. We avoid this problem by directly computing the exact FIM.

**Algorithm 1** Natural Policy Gradient Method with Parameter-based Exploration

---
**Require:** $\theta = \langle \mathbf{w}, \mathbf{C} \rangle$: policy parameters, $\psi(\mathbf{u}|\mathbf{s}, \mathbf{w})$: controller, $\alpha$: step size, $\beta$: discount rate, $b$: baseline.
1: Initialize $\tilde{\mathbf{z}}_0 = \mathbf{0}$, observe $\mathbf{s}_1$.
2: **for** $t = 1, ...$ **do**
3:      Draw $\xi_t \sim \mathcal{N}(\mathbf{0}, \mathbf{I})$, compute action $\tilde{\mathbf{w}}_t = \mathbf{C}^{\mathrm{T}}\xi_t + \mathbf{w}$.
4:      Execute $\mathbf{u}_t \sim \psi(\mathbf{u}_t|\mathbf{s}_t, \tilde{\mathbf{w}}_t)$, obtain observation $\mathbf{s}_{t+1}$ and reward $r_t$.
5:      $\tilde{\nabla}_{\mathbf{w}} \log \mu(\tilde{\mathbf{w}}_t|\theta) = \tilde{\mathbf{w}}_t - \mathbf{w}$, $\tilde{\nabla}_{\mathbf{C}} \log \mu(\tilde{\mathbf{w}}_t|\theta) = \{\mathrm{triu}(\xi_t\xi_t^{\mathrm{T}}) - \frac{1}{2}\mathrm{diag}(\xi_t\xi_t^{\mathrm{T}}) - \frac{1}{2}\mathbf{I}\}\mathbf{C}$
6:      $\tilde{\mathbf{z}}_t = \beta\tilde{\mathbf{z}}_{t-1} + \tilde{\nabla}_\theta \log \mu(\tilde{\mathbf{w}}_t|\theta)$
7:      $\theta \leftarrow \theta + \alpha(r_t - b)\tilde{\mathbf{z}}_t$
8: **end for**

---

Substituting $\pi = \mu(\tilde{\mathbf{w}}|\theta)$ into (1), we can rewrite the policy gradient to obtain

$$\nabla_\theta \eta(\theta) = \int_{\mathcal{S}} p_D(\mathbf{s}|\theta) \int_{\mathcal{W}} \mu(\tilde{\mathbf{w}}|\theta)\nabla_\theta \log \mu(\tilde{\mathbf{w}}|\theta)Q_\theta(\mathbf{s}, \tilde{\mathbf{w}})d\tilde{\mathbf{w}}d\mathbf{s}.$$

Furthermore, the FIM of this distribution is

$$\mathbf{F}_\theta = \int_{\mathcal{S}} p_D(\mathbf{s}|\theta) \int_{\mathcal{W}} \mu(\tilde{\mathbf{w}}|\theta)\nabla_\theta \log \mu(\tilde{\mathbf{w}}|\theta)\nabla_\theta \log \mu(\tilde{\mathbf{w}}|\theta)^{\mathrm{T}}d\tilde{\mathbf{w}}d\mathbf{s}$$

$$= \int_{\mathcal{W}} \mu(\tilde{\mathbf{w}}|\theta)\nabla_\theta \log \mu(\tilde{\mathbf{w}}|\theta)\nabla_\theta \log \mu(\tilde{\mathbf{w}}|\theta)^{\mathrm{T}}d\tilde{\mathbf{w}}.$$

Because $\mathbf{F}_\theta$ is independent of $p_D(\mathbf{s}|\theta)$, we can use the real FIM.

Sun et al. [19] proved that the precise FIM of the Gaussian distribution $\mathcal{N}(\mathbf{w}, \mathbf{C}^{\mathrm{T}}\mathbf{C})$ becomes a block-diagonal matrix $\mathrm{diag}(\mathbf{F}_0, ..., \mathbf{F}_n)$ whose first block $\mathbf{F}_0$ is identical to $\tilde{\Sigma}^{-1}$ and whose $k$-th ($1 \le k \le n$) block $\mathbf{F}_k$ is given by

$$\mathbf{F}_k = \begin{bmatrix} c_{k,k}^{-2} & \mathbf{0} \\ \mathbf{0} & \mathbf{0} \end{bmatrix} + \tilde{\Sigma}^{-1}_{k:n,k:n}$$

$$= [\mathbf{0} \quad \mathbf{I}_{\bar{k}}] \mathbf{C}^{-1} (\mathbf{v}_k\mathbf{v}_k^{\mathrm{T}} + \mathbf{I}) \mathbf{C}^{-\mathrm{T}} \begin{bmatrix} \mathbf{0} \\ \mathbf{I}_{\bar{k}} \end{bmatrix},$$

where $\mathbf{v}_k$ denotes an $n$-dimensional column vector of which the only nonzero element is the $k$-th element that is one, and $\mathbf{I}_{\bar{k}}$ is the $[n - k + 1]$-dimensional identity matrix.

Further, Akimoto et al. [1] derived the inverse matrix of the $k$-th diagonal block $\mathbf{F}_k$ of the FIM. Because $\mathbf{F}_\theta$ is a block-diagonal matrix and $\mathbf{C}$ is upper triangular, it is easy to verify that the inverse matrix of the FIM is

$$\mathbf{F}_k^{-1} = [\mathbf{0} \quad \mathbf{I}_{\bar{k}}] \mathbf{C}^{\mathrm{T}} \left( -\frac{1}{2}\mathbf{v}_k\mathbf{v}_k^{\mathrm{T}} + \begin{bmatrix} \mathbf{0} & \mathbf{0} \\ \mathbf{0} & \mathbf{I}_{\bar{k}} \end{bmatrix} \right) \mathbf{C} \begin{bmatrix} \mathbf{0} \\ \mathbf{I}_{\bar{k}} \end{bmatrix},$$

where we use

$$\mathbf{v}_k^{\mathrm{T}}\mathbf{C} \begin{bmatrix} \mathbf{0} & \mathbf{0} \\ \mathbf{0} & \mathbf{I}_{\bar{k}} \end{bmatrix} \mathbf{C}^{-1} = \mathbf{v}_k^{\mathrm{T}} \text{ and } [\mathbf{0} \quad \mathbf{I}_{\bar{k}}] \mathbf{C} \begin{bmatrix} \mathbf{0} & \mathbf{0} \\ \mathbf{0} & \mathbf{I}_{\bar{k}} \end{bmatrix} \mathbf{C}^{-1} = [\mathbf{0} \quad \mathbf{I}_{\bar{k}}]. \tag{4}$$

### 3.3 Natural Policy Gradient

Now, we derive the eligibility premultiplied by the inverse matrix of the FIM $\tilde{\nabla}_\theta \log \mu(\tilde{\mathbf{w}}_t|\theta) = \mathbf{F}_\theta^{-1}\nabla_\theta \log \mu(\tilde{\mathbf{w}}_t|\theta)$ in the same manner as [1]. The characteristic eligibility w.r.t. $\mathbf{w}$ is given by

$$\nabla_{\mathbf{w}} \log \mu(\tilde{\mathbf{w}}_t|\theta) = \tilde{\Sigma}^{-1}(\tilde{\mathbf{w}}_t - \mathbf{w}).$$

Obviously, $\mathbf{F}_0^{-1} = \tilde{\Sigma}$ and $\tilde{\nabla}_{\mathbf{w}} \log \mu(\tilde{\mathbf{w}}_t|\theta) = \mathbf{F}_0^{-1}\nabla_{\mathbf{w}} \log \mu(\tilde{\mathbf{w}}_t|\theta) = \tilde{\mathbf{w}}_t - \mathbf{w}$. The characteristic eligibility w.r.t. $\mathbf{C}$ is given by

$$\frac{\partial}{\partial c_{i,j}} \log \mu(\tilde{\mathbf{w}}_t|\theta) = \mathbf{v}_i^{\mathrm{T}} \left( \mathrm{triu}(\mathbf{Y}_t\mathbf{C}^{-\mathrm{T}}) - \mathrm{diag}(\mathbf{C}^{-1}) \right) \mathbf{v}_j,$$

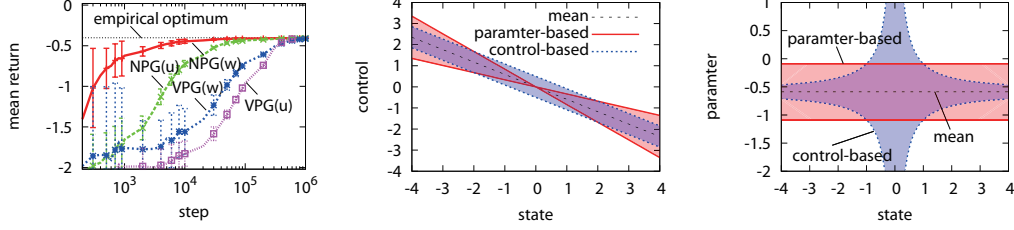

Figure 2: Performance of NPG(w) as compared to that of NPG(u), VPG(w), and VPG(u) in the linear quadratic regulation task averaged over 100 trials. Left: The empirical optimum denotes the mean return under the optimum gain. Center and Right: Illustration of the main difference between control- and parameter-based exploration. The sampling area of $1\sigma$ in the state-control space (center) and the state-parameter space (right) is plotted.

where $\mathrm{triu}(\mathbf{Y}_t\mathbf{C}^{-\mathrm{T}})$ denotes the upper triangular matrix whose $(i,j)$ element is identical to the $(i,j)$ element of $\mathbf{Y}_t\mathbf{C}^{-\mathrm{T}}$ if $i \le j$ and zero otherwise, and $\mathbf{Y}_t = \mathbf{C}^{-\mathrm{T}}(\tilde{\mathbf{w}}_t - \mathbf{w})(\tilde{\mathbf{w}}_t - \mathbf{w})^{\mathrm{T}}\mathbf{C}^{-1}$ is a symmetric matrix.

Let $\mathbf{c}_k = (c_{k,k}, ..., c_{k,n})^{\mathrm{T}}$ (of dimension $n + 1 - k$); then, the characteristic eligibility w.r.t. $\mathbf{c}_k$ is expressed as

$$\nabla_{\mathbf{c}_k} \log \mu(\tilde{\mathbf{w}}_t|\theta) = [\mathbf{0} \quad \mathbf{I}_{\bar{k}}] \left( \mathbf{C}^{-1}\mathbf{Y}_t - \mathrm{diag}(\mathbf{C}^{-1}) \right) \mathbf{v}_k.$$

According to (4), $\mathrm{diag}(\mathbf{C}^{-1})\mathbf{v}_k = c_{k,k}^{-1}\mathbf{v}_k$ and

$$\mathbf{v}_k^{\mathrm{T}}\mathbf{C}\mathbf{v}_k = c_{k,k} \quad \text{and} \quad \begin{bmatrix} \mathbf{0} & \mathbf{0} \\ \mathbf{0} & \mathbf{I}_{\bar{k}} \end{bmatrix} \mathbf{C}\mathbf{v}_k = c_{k,k}\mathbf{v}_k,$$

the $k$-th block of $\mathbf{F}_\theta^{-1}\nabla_\theta \log \mu(\tilde{\mathbf{w}}_t|\theta)$ is therefore

$$\tilde{\nabla}_{\mathbf{c}_k} \log \mu(\tilde{\mathbf{w}}_t|\theta) = \mathbf{F}_k^{-1}\nabla_{\mathbf{c}_k} \log \mu(\tilde{\mathbf{w}}_t|\theta)$$

$$= [\mathbf{0} \quad \mathbf{I}_{\bar{k}}] \mathbf{C}^{\mathrm{T}} \left( -\frac{1}{2}\mathbf{v}_k\mathbf{v}_k^{\mathrm{T}} + \begin{bmatrix} \mathbf{0} & \mathbf{0} \\ \mathbf{0} & \mathbf{I}_{\bar{k}} \end{bmatrix} \right) \mathbf{C} \begin{bmatrix} \mathbf{0} & \mathbf{0} \\ \mathbf{0} & \mathbf{I}_{\bar{k}} \end{bmatrix} \left( \mathbf{C}^{-1}\mathbf{Y}_t - \mathrm{diag}(\mathbf{C}^{-1}) \right) \mathbf{v}_k$$

$$= [\mathbf{0} \quad \mathbf{I}_{\bar{k}}] \mathbf{C}^{\mathrm{T}} \left( -\frac{1}{2}\mathbf{v}_k\mathbf{v}_k^{\mathrm{T}} + \begin{bmatrix} \mathbf{0} & \mathbf{0} \\ \mathbf{0} & \mathbf{I}_{\bar{k}} \end{bmatrix} \right) (\mathbf{Y}_t - \mathbf{I})\mathbf{v}_k.$$

Because $\tilde{\nabla}_{\mathbf{c}_k} \log \mu(\tilde{\mathbf{w}}_t|\theta)^{\mathrm{T}} = \left( \tilde{\nabla}_{\mathbf{C}} \log \mu(\tilde{\mathbf{w}}_t|\theta) \right)_{k,k:n}$, we obtain

$$\tilde{\nabla}_{\mathbf{C}} \log \mu(\tilde{\mathbf{w}}_t|\theta) = \left( \mathrm{triu}(\mathbf{Y}_t) - \frac{1}{2}\mathrm{diag}(\mathbf{Y}_t) - \frac{1}{2}\mathbf{I} \right) \mathbf{C}. \tag{5}$$

Therefore, the time complexity of computing

$$\tilde{\nabla}_\theta \log \mu(\tilde{\mathbf{w}}_t|\theta) = [\tilde{\nabla}_{\mathbf{w}} \log \mu(\tilde{\mathbf{w}}_t|\theta)^{\mathrm{T}}, \tilde{\nabla}_{\mathbf{c}_1} \log \mu(\tilde{\mathbf{w}}_t|\theta)^{\mathrm{T}}, ..., \tilde{\nabla}_{\mathbf{c}_n} \log \mu(\tilde{\mathbf{w}}_t|\theta)^{\mathrm{T}}]^{\mathrm{T}}$$

is $O(n^3)$, which is of the same order as the computation of $\nabla_\theta \log \mu(\tilde{\mathbf{w}}_t|\theta)$. This is a significant improvement over the current natural policy gradient estimation using (2) and (3) with parameter-based exploration, whose complexity is $O(n^6)$. Note that more simple forms for exploration distribution could be used. When we use the exploration strategy that is represented as an independent normal distribution for each parameter $\mathbf{w}_i$ in $\mathbf{w}$, the natural policy gradient is estimated in $O(n)$ time. This limited form ignores the relationship between parameters, but it is practical for high-dimensional controllers.

### 3.4 An Algorithm

For a parameterized class of controllers $\psi(\mathbf{u}|\mathbf{s}, \mathbf{w})$, we can use the exploration strategy $\mu(\tilde{\mathbf{w}}|\theta)$. An online version based on the GPOMDP algorithm of this implementation is shown in Algorithm 1. In practice, the parameters of the controller $\tilde{\mathbf{w}}_t$ are generated by $\tilde{\mathbf{w}}_t = \mathbf{C}^{\mathrm{T}}\xi_t + \mathbf{w}$, where $\xi_t \sim \mathcal{N}(\mathbf{0}, \mathbf{I})$ are normal random numbers. Now, we can instead use $\mathbf{Y}_t = \mathbf{C}^{-\mathrm{T}}(\tilde{\mathbf{w}}_t - \mathbf{w})(\tilde{\mathbf{w}}_t - \mathbf{w})^{\mathrm{T}}\mathbf{C}^{-1} = \xi_t\xi_t^{\mathrm{T}}$. To reduce the variance of the gradient estimation, we employ variance reduction techniques [6] to adapt the reinforcement baseline $b$.

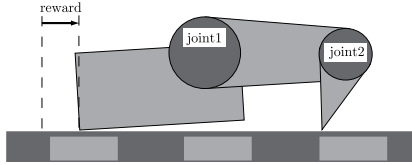

Figure 3: Simulator of a two-link arm robot.

## 4 Experiments

In this section, we evaluate the performance of our proposed NPGPE method. The efficiency of parameter-based exploration has been reported for episodic tasks [18]. We compare parameter- and control-based exploration strategies with natural gradient and conventional "vanilla" gradients using a simple continuing task as an example of a linear control problem. We also demonstrate NPGPE's usefulness for a physically realistic locomotion task using a two-link arm robot simulator.

### 4.1 Implementation

We compare two different exploration strategies. The first is the parameter-based exploration strategy $\mu(\tilde{\mathbf{w}}|\theta)$ presented in Section 3.1. The second is the control-based exploration strategy $\epsilon(\mathbf{u}|\tilde{\mathbf{u}}, \mathbf{D})$ represented by a normal distribution for a control space, where $\tilde{\mathbf{u}}$ is the mean vector of the control generated by controller $\psi$ and $\mathbf{D}$ represents the Cholesky decomposition of the covariance matrix $\Sigma$ such that $\mathbf{D}$ is an $m \times m$ upper triangular matrix and $\Sigma = \mathbf{D}^{\mathsf{T}}\mathbf{D}$. The parameters of the policy $\pi_U(\mathbf{u}|\mathbf{s}, \theta)$ are $\theta = \langle \mathbf{w}, \mathbf{D} \rangle$ to be an $[n + m(m+1)/2]$-dimensional column vector consisting of the elements of $\mathbf{w}$ and the upper-right elements of $\mathbf{D}$.

### 4.2 Linear Quadratic Regulator

The following linear control problem can serve as a benchmark of delayed reinforcement tasks [10]. The dynamics of the environment is

$$\mathbf{s}_{t+1} = \mathbf{s}_t + \mathbf{u}_t + \delta,$$

where $\mathbf{s} \in \Re^1$, $\mathbf{u} \in \Re^1$, and $\delta \sim \mathcal{N}(0, 0.5^2)$. The immediate reward is given by $r_t = -\mathbf{s}_t^2 - \mathbf{u}_t^2$. In this experiment, the set of possible states is constrained to lie in the range [-4, 4], and $\mathbf{s}_t$ is truncated. When the agent chooses an action that does not lie in the range $[-4, 4]$, the action executed in the environment is also truncated. The controller is represented by $\psi(\mathbf{u}|\mathbf{s}, \mathbf{w}) = \mathbf{s} \cdot \mathbf{w}$, where $\mathbf{w} \in \Re^1$. The optimal parameter is given by $\mathbf{w}^* = 2/(1 + 2\beta + \sqrt{4\beta^2 + 1}) - 1$ from the Riccati equation.

For clarification, we now write an NPG that employs the natural policy gradient and a VPG that employs the "vanilla" policy gradient. Therefore, NPG(w) and VPG(w) denote the use of the parameter-based exploration strategy, and NPG(u) and VPG(u) denote the use of the control-based exploration strategy. Our proposed NPGPE method is NPG(w).

Figure2 (left) shows the performance of all compared methods. We can see that the algorithm using parameter-based exploration had better performance than that using control-based exploration in the continuing task. The natural policy gradient also improved the convergence speed, and a combination with parameter-based exploration outperformed all other methods. The reason for the acceleration in learning in this case may be the fact that the samples generated by the parameter-based exploration strategy allow effective search. Figure2 (center and right) show plots of the sampling area in the state-control space and the state-parameter space, respectively. Because control-based exploration maintains the sampling area in the control space, the sampling is almost uniform in the parameter space at around $\mathbf{s} = 0$, where the agent visits frequently. Therefore, the parameter-based exploration may realize more efficient sampling than the control-based exploration.

### 4.3 Locomotion Task on a Two-link Arm Robot

We applied the algorithm to the robot shown in Figure3 of Kimura et al. [11]. The objective of learning is to find control rules to move forward. The joints are controlled by servo motors that react

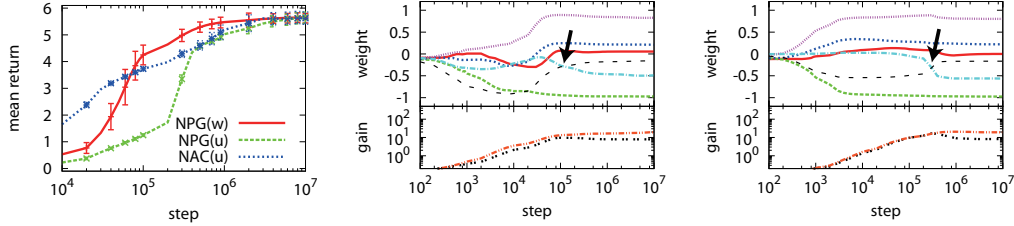

Figure 4: Performance of NPG(w) as compared to that of NPG(u) and NAC(u) in the locomotion task averaged over 100 trials. Left: Mean performance of all compared methods. Center: Parameters of controller for NPG(w). Right: Parameters of controller for NPG(u). The parameters of the controller are normalized by $gain_i = \sqrt{\sum_j \mathbf{w}_{i,j}}$ and $weight_{i,j} = \mathbf{w}_{i,j}/gain_i$, where $\mathbf{w}_{i,j}$ denotes the $j$-th parameter of the $i$-th joint. Arrows in the center and right denote the changing points of the relation between two important parameters.

to angular-position commands. At each time step, the agent observes the angular position of two motors, where each observation $o_1, o_2$ is normalized to $[0, 1]$, and selects an action. The immediate reward is the distance of the body movement caused by the previous action. When the robot moves backward, the agent receives a negative reward. The state vector is expressed as $\mathbf{s} = [o_1, o_2, 1]^{\mathrm{T}}$. The control for motor $i$ is generated by $\mathbf{u}_i = 1/(1 + \exp(-\sum_j \mathbf{s}_j \mathbf{w}_{i,j}))$. The dimension of the parameters of the policies is $d_W = n(n + 3)/2 = 27$ and $d_U = n + m(m + 1)/2 = 9$ for the parameter- and control-based exploration strategy, respectively.

We compared NPG(w), i.e., NPGPE, with NPG(u) and NAC(u). NAC is the state-of-the-art policy gradient algorithm [15] that combines natural policy gradients, actor-critic framework, and least-squares temporal-difference Q-learning. NAC computes the inverse of a $d \times d$ matrix to estimate the natural steepest ascent direction. Because NAC(w) has $O(d_W^3)$ time complexity for each iteration, which is prohibitively expensive, we apply NAC to only control-based exploration.

Figure4 (left) shows our results. Initially, NPG(w) is outperformed by NAC(u); however, it then reaches good solutions with fewer steps. Furthermore, at a later stage, NAC(u) matches NPG(u). Figure4 (center and right) show the path of the relation between the parameters of the controller. NPG(w) is much slower than NPG(u) to adapt the relation at an early stage; however, it can seek the relations of important parameters (indicated by arrows in the figures) faster, whereas NPG(u) gets stuck because of inefficient sampling.

# 5   Conclusions

This paper proposed a novel natural policy gradient method combined with parameter-based exploration to cope with high-dimensional reinforcement learning domains. The proposed algorithm, NPGPE, is very simple and quickly calculates the estimation of the natural policy gradient. Moreover, the experimental results demonstrate a significant improvement in the control domain.

Future works will focus on developing actor-critic versions of NPGPE that might encourage performance improvements at an early stage, and on combining other gradient methods such as natural conjugate gradient methods [8].

In addition, a comparison with other direct parameter perturbation methods such as finite difference gradient methods [14], CMA-ES [7], and NES [19] will be necessary to gain a better understanding of the properties and efficacy of the combination of parameter-based exploration strategies and the natural policy gradient. Furthermore, the application of the algorithm to real-world problems is required to assess its utility.

**Acknowledgments**

This work was suported by the Japan Society for the Promotion of Science (22 9031).

## Footnotes

[1][5] showed that the approximation error is proportional to $(1-\beta)/(1-|\kappa_2|)$, where $\kappa_2$ is the sub-dominant eigenvalue of the Markov chain

# References

[1] Youhei Akimoto, Yuichi Nagata, Isao Ono, and Shigenobu Kobayashi. Bidirectional Relation between CMA Evolution Strategies and Natural Evolution Strategies. *Parallel Problem Solving from Nature XI*, pages 154–163, 2010.

[2] S. Amari. Natural Gradient Works Efficiently in Learning. *Neural Computation*, 10(2):251–276, 1998.

[3] S. Amari and H. Nagaoka. *Methods of Information Geometry*. American Mathematical Society, 2007.

[4] J. Andrew Bagnell and Jeff Schneider. Covariant policy search. In *IJCAI'03: Proceedings of the 18th international joint conference on Artificial intelligence*, pages 1019–1024, 2003.

[5] Jonathan Baxter and Peter L. Bartlett. Infinite-horizon policy-gradient estimation. *Journal of Artificial Intelligence Research*, 15:319–350, 2001.

[6] Evan Greensmith, Peter L. Bartlett, and Jonathan Baxter. Variance reduction techniques for gradient estimates in reinforcement learning. *The Journal of Machine Learning Research*, 5:1471–1530, 2004.

[7] V. Heidrich-Meisner and C. Igel. Variable metric reinforcement learning methods applied to the noisy mountain car problem. In *EWRL 2008*, pages 136–150, 2008.

[8] Antti Honkela, Matti Tornio, Tapani Raiko, and Juha Karhunen. Natural conjugate gradient in variational inference. In *ICONIP 2007*, pages 305–314, 2008.

[9] S. A. Kakade. A natural policy gradient. In *In Advances in Neural Information Processing Systems*, pages 1531–1538, 2001.

[10] H. Kimura and S. Kobayashi. Reinforcement learning for continuous action using stochastic gradient ascent. In *Intelligent Autonomous Systems (IAS-5)*, pages 288–295, 1998.

[11] Hajime Kimura, Kazuteru Miyazaki, and Shigenobu Kobayashi. Reinforcement learning in pomdps with function approximation. In *ICML '97: Proceedings of the Fourteenth International Conference on Machine Learning*, pages 152–160, 1997.

[12] Hajime Kimura, Masayuki Yamamura, and Shigenobu Kobayashi. Reinforcement learning by stochastic hill climbing on discounted reward. In *ICML*, pages 295–303, 1995.

[13] Jens Kober and Jan Peters. Policy search for motor primitives in robotics. In *Advances in Neural Information Processing Systems 21*, pages 849–856, 2009.

[14] Jan Peters and Stefan Schaal. Policy Gradient Methods for Robotics. In *2006 IEEE/RSJ International Conference on Intelligent Robots and Systems*, pages 2219–2225, 2006.

[15] Jan Peters and Stefan Schaal. Natural actor-critic. *Neurocomputing*, 71(7–9):1180–1190, 2008.

[16] Silvia Richter, Douglas Aberdeen, and Jin Yu. Natural actor-critic for road traffic optimisation. In *Advances in Neural Information Processing Systems 19*, pages 1169–1176. MIT Press, Cambridge, MA, 2007.

[17] Thomas Rückstieß, Martin Felder, and Jürgen Schmidhuber. State-dependent exploration for policy gradient methods. In *ECML PKDD '08: Proceedings of the European conference on Machine Learning and Knowledge Discovery in Databases - Part II*, pages 234–249, 2008.

[18] Frank Sehnke, C Osendorfer, T Rueckstiess, A. Graves, J. Peters, and J. Schmidhuber. Policy gradients with parameter-based exploration for control. In *Proceedings of the International Conference on Artificial Neural Networks (ICANN)*, pages 387–396, 2008.

[19] Yi Sun, Daan Wierstra, Tom Schaul, and Juergen Schmidhuber. Efficient natural evolution strategies. In *GECCO '09: Proceedings of the 11th Annual conference on Genetic and evolutionary computation*, pages 539–546, 2009.

[20] R. S. Sutton. Policy gradient method for reinforcement learning with function approximation. In *Advances in Neural Information Processing Systems*, volume 12, pages 1057–1063, 2000.

[21] Daan Wierstra, Er Foerster, Jan Peters, and Juergen Schmidhuber. Solving deep memory pomdps with recurrent policy gradients. In *In International Conference on Artificial Neural Networks*, 2007.

[22] Ronald J. Williams. Simple statistical gradient-following algorithms for connectionist reinforcement learning. In *Machine Learning*, pages 229–256, 1992.

